# Scale Mixtures of Gaussians and the Statistics of Natural Images

**Martin J. Wainwright**
Stochastic Systems Group
Electrical Engineering & CS
MIT, Building 35-425
Cambridge, MA 02139
*mjwain@mit.edu*

**Eero P. Simoncelli**
Ctr. for Neural Science, and
Courant Inst. of Mathematical Sciences
New York University
New York, NY 10012
*eero.simoncelli@nyu.edu*

## Abstract

The statistics of photographic images, when represented using multiscale (wavelet) bases, exhibit two striking types of non-Gaussian behavior. First, the marginal densities of the coefficients have extended heavy tails. Second, the joint densities exhibit variance dependencies not captured by second-order models. We examine properties of the class of Gaussian scale mixtures, and show that these densities can accurately characterize both the marginal and joint distributions of natural image wavelet coefficients. This class of model suggests a Markov structure, in which wavelet coefficients are linked by hidden scaling variables corresponding to local image structure. We derive an estimator for these hidden variables, and show that a nonlinear "normalization" procedure can be used to Gaussianize the coefficients.

Recent years have witnessed a surge of interest in modeling the statistics of natural images. Such models are important for applications in image processing and computer vision, where many techniques rely (either implicitly or explicitly) on a prior density. A number of empirical studies have demonstrated that the power spectra of natural images follow a $1/f^\gamma$ law in radial frequency, where the exponent $\gamma$ is typically close to two [e.g., 1]. Such second-order characterization is inadequate, however, because images usually exhibit highly non-Gaussian behavior. For instance, the marginals of wavelet coefficients typically have much heavier tails than a Gaussian [2]. Furthermore, despite being approximately decorrelated (as suggested by theoretical analysis of $1/f$ processes [3]), orthonormal wavelet coefficients exhibit striking forms of statistical dependency [4, 5]. In particular, the standard deviation of a wavelet coefficient typically scales with the absolute values of its neighbors [5].

A number of researchers have modeled the marginal distributions of wavelet coefficients with generalized Laplacians, $p_Y(y) \propto \exp(-|y/\lambda|^p)$ [e.g. 6, 7, 8]. Special cases include the Gaussian ($p = 2$) and the Laplacian ($p = 1$), but appropriate ex-

Research supported by NSERC 1969 fellowship 160833 to MJW, and NSF CAREER grant MIP-9796040 to EPS.

| Mixing density | GSM density | GSM char. function |
|---|---|---|
| $\sqrt{Z(\gamma)}$ | symmetrized Gamma | $(1 + \frac{t^2}{2\lambda^2})^{-\gamma}, \quad \gamma > 0$ |
| $1/\sqrt{Z(\beta - \frac{1}{2})}$ | Student: $[1/(\lambda^2 + y^2)]^\beta, \quad \beta > \frac{1}{2}$ | No explicit form |
| Positive, $\sqrt{\frac{\alpha}{2}}$ − stable | $\alpha$-stable | $\exp{(-|\lambda t|^\alpha)}, \quad \alpha \in (0, 2]$ |
| No explicit form | generalized Laplacian: $\exp{(-|y/\lambda|^p)}, \quad p \in (0, 2]$ | No explicit form |

**Table 1.** Example densities from the class of Gaussian scale mixtures. $Z(\gamma)$ denotes a positive gamma variable, with density $p(z) = [1/\Gamma(\gamma)] z^{\gamma-1} \exp{(-z)}$. The characteristic function of a random variable $x$ is defined as $\phi_x(t) \triangleq \int_{-\infty}^{\infty} p(x) \exp{(jxt)} dx$.

ponents for natural images are typically less than one. Simoncelli [5, 9] has modeled the variance dependencies of pairs of wavelet coefficients. Romberg et al. [10] have modeled wavelet densities using two-component mixtures of Gaussians. Huang and Mumford [11] have modeled marginal densities and cross-sections of joint densities with multi-dimensional generalized Laplacians.

In the following sections, we explore the semi-parametric class of *Gaussian scale mixtures*. We show that members of this class satisfy the dual requirements of being heavy-tailed, and exhibiting multiplicative scaling between coefficients. We also show that a particular member of this class, in which the multiplier variables are distributed according to a gamma density, captures the range of joint statistical behaviors seen in wavelet coefficients of natural images. We derive an estimator for the multipliers, and show that a nonlinear "normalization" procedure can be used to Gaussianize the wavelet coefficients. Lastly, we form random cascades by linking the multipliers on a multiresolution tree.

## 1   Scale Mixtures of Gaussians

A random vector $Y$ is a Gaussian scale mixture (GSM) if $Y \overset{d}{=} zU$, where $\overset{d}{=}$ denotes equality in distribution; $z \geq 0$ is a scalar random variable; $U \sim \mathcal{N}(0, Q)$ is a Gaussian random vector; and $z$ and $U$ are independent.

As a consequence, any GSM variable has a density given by an integral:

$$p_Y(Y) = \int_{-\infty}^{\infty} \frac{1}{[2\pi]^{\frac{N}{2}} |z^2 Q|^{1/2}} \exp\left(-\frac{Y^T Q^{-1} Y}{2z^2}\right) \phi_z(z) dz.$$

where $\phi_z$ is the probability density of the mixing variable $z$ (henceforth the multiplier). A special case of a GSM is a finite mixture of Gaussians, where $z$ is a discrete random variable. More generally, it is straightforward to provide conditions on either the density [12] or characteristic function of $X$ that ensure it is a GSM, but these conditions do not necessarily provide an explicit form of $\phi_z$. Nevertheless, a number of well-known distributions may be written as Gaussian scale mixtures. For the scalar case, a few of these densities, along with their associated characteristic functions, are listed in Table 1. Each variable is characterized by a scale parameter $\lambda$, and a tail parameter. All of the GSM models listed in Table 1 produce heavy-tailed marginal and variance-scaling joint densities.

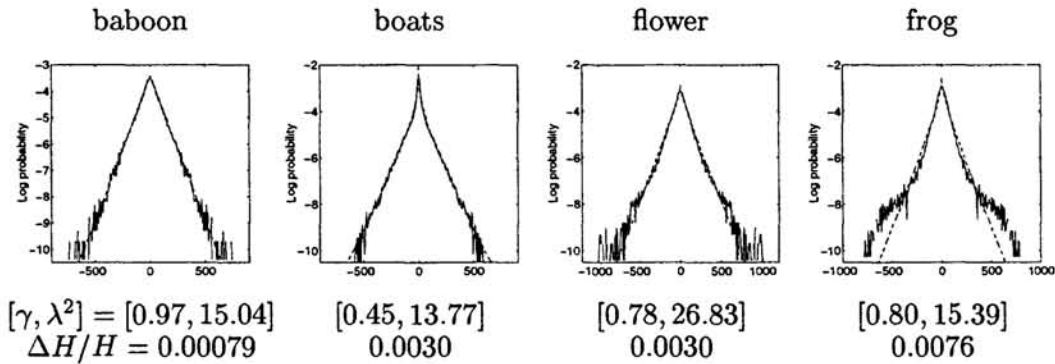

| baboon | boats | flower | frog |

$[\gamma, \lambda^2] = [0.97, 15.04]$   $[0.45, 13.77]$   $[0.78, 26.83]$   $[0.80, 15.39]$
$\Delta H/H = 0.00079$            $0.0030$          $0.0030$          $0.0076$

**Figure 1.** GSMs (dashed lines) fitted to empirical histograms (solid lines). Below each plot are the parameter values, and the relative entropy between the histogram (with 256 bins) and the model, as a fraction of the histogram entropy.

## 2   Modeling Natural Images

As mentioned in the introduction, natural images exhibit striking non-Gaussian behavior, both in their marginal and joint statistics. In this section, we show that this behavior is consistent with a GSM, using the first of the densities given in Table 1 for illustration.

### 2.1   Marginal distributions

We begin by examining the symmetrized Gamma class as a model for marginal distributions of wavelet coefficients. Figure 1 shows empirical histograms of a particular wavelet subband[1] for four different natural images, along with the best fitting instance of the symmetrized Gamma distribution. Fitting was performed by minimizing the relative entropy (i.e., the Kullback-Leibler divergence, denoted $\Delta H$) between empirical and theoretical histograms. In general, the fits are quite good: the fourth plot shows one of the worst fits in our data set.

### 2.2   Normalized components

For a GSM random vector $Y \overset{d}{=} zU$, the normalized variable $Y/z$ formed by component-wise division is Gaussian-distributed. In order to test this behavior empirically, we model a given wavelet coefficient $y_0$ and a collection of neighbors $\{y_1, \ldots, y_N\}$ as a GSM vector. For our examples, we use a neighborhood of $N = 11$ coefficients corresponding to basis functions at 4 adjacent positions, 5 orientations, and 2 scales. Although the multiplier $z$ is unknown, we can estimate it by maximizing the log likelihood of the observed coefficients: $\hat{z} \overset{\triangle}{=} \arg\max_z \{\log p(Y|z)\}$. Under reasonable conditions, the normalized quantity $Y/\hat{z}$ should converge in distribution to a Gaussian as the number of neighbors increases. The estimate $\hat{z}$ is simple to derive:

$$
\begin{aligned}
\hat{z} &= \arg\max_z \left\{\log p(Y|z)\right\} \\
&= \arg\min_z \left\{N \log(z) + Y^T Q^{-1} Y / 2z^2\right\} \\
&= \sqrt{Y^T Q^{-1} Y / N},
\end{aligned}
$$

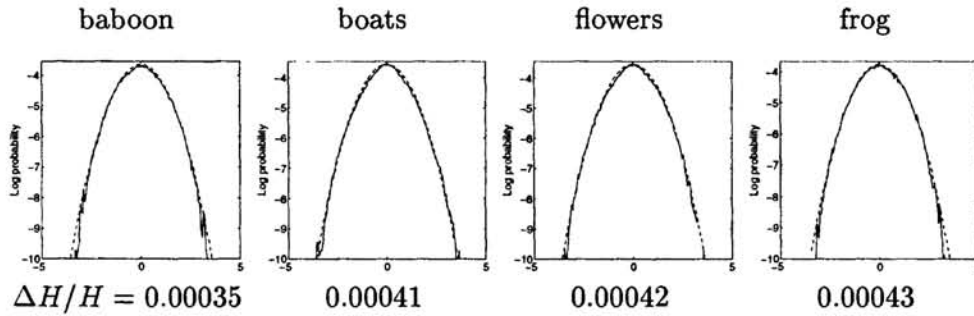

**Figure 2.** Marginal log histograms (solid lines) of the normalized coefficient $\nu$ for a single subband of four natural images. Each shape is close to an inverted parabola, in agreement with Gaussians (dashed lines) of equivalent empirical variance. Below each plot is the relative entropy between the histogram (with 256 bins) and a variance-matched Gaussian, as a fraction of the total histogram entropy.

where $Q \triangleq \mathbb{E}\left[UU^T\right]$ is the positive definite covariance matrix of the underlying Gaussian vector $U$.

Given the estimate $\hat{z}$, we then compute the normalized coefficient $\nu \triangleq y_0/\hat{z}$. This is a generalization of the variance normalization proposed by Ruderman and Bialek[1], and the weighted sum of squares normalization procedure used by Simoncelli [5, 14]. Figure 2 shows the marginal histograms (in the log domain) of this normalized coefficient for four natural images, along with Gaussians of equal empirical variance. In contrast to histograms of the raw coefficients (shown in Figure 1), the histograms of normalized coefficients are nearly Gaussian.

The GSM model makes a stronger prediction: that normalized quantities corresponding to nearby wavelet pairs should be *jointly* Gaussian. Specifically, a pair of normalized coefficients should be either correlated or uncorrelated Gaussians, depending on whether the underlying Gaussians $U = [u_1 \, u_2]^T$ are correlated or uncorrelated. We examine this prediction by collecting joint conditional histograms of normalized coefficients. The top row of Figure 3 shows joint conditional histograms for raw wavelet coefficients (taken from the same four natural images as Figure 2). The first two columns correspond to adjacent spatial scales; though decorrelated, they exhibit the familiar form of multiplicative scaling. The latter two columns correspond to adjacent orientations; in addition to being correlated, they also exhibit the multiplicative form of dependency.

The bottom row shows the same joint conditional histograms, after the coefficients have been normalized. Whereas Figure 2 demonstrates that normalized coefficients are close to *marginally* Gaussian, Figure 3 demonstrates that they are also approximately *jointly* Gaussian. These observations support the use of a Gaussian scale mixture for modeling natural images.

## 2.3   Joint distributions

The GSM model is a reasonable approximation for groups of nearby wavelet coefficients. However, the components of GSM vectors are highly dependent, whereas the dependency between wavelet coefficients decreases as (for example) their spatial separation increases. Consequently, the simple GSM model is inadequate for global modeling of coefficients. We are thus led to use a graphical model (such as tree) that specifies probabilistic relations between the multipliers. The wavelet coefficients themselves are considered observations, and are linked indirectly by their shared dependency on the (hidden) multipliers.

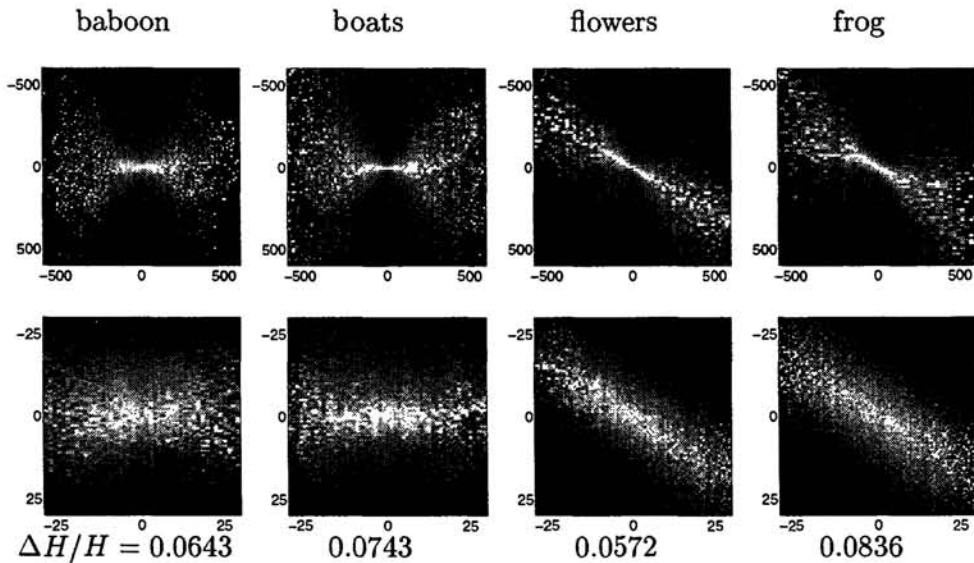

**Figure 3.** Top row: joint conditional histograms of raw wavelet coefficients for four natural images. Bottom row: joint conditional histograms of normalized pairs of coefficients. Below each plot is the relative entropy between the joint histogram (with $256 \times 256$ bins) and a covariance-matched Gaussian, as a fraction of the total histogram entropy.

For concreteness, we model the wavelet coefficient at node $s$ as $y(s) \overset{d}{=} \|x(s)\| \, u(s)$, where $x(s)$ is Gaussian, so that $z \triangleq \|x\|$ is the square root of a gamma variable of index 0.5. For illustration, we assume that the multipliers are linked by a multiscale autoregressive (MAR) process [15] on a tree:

$$x(s) = \mu \, x(p(s)) + \sqrt{1 - \mu^2} \, w(s)$$

where $p(s)$ is the parent of node $s$. Two wavelet coefficients $y(s)$ and $y(t)$ are linked through the multiplier at their common ancestral node denoted $s \wedge t$. In particular, the joint distributions are given by

$$y(s) = \left\| \mu^{d(s, s \wedge t)} \, x(s \wedge t) + v_1(s) \right\| u(s)$$
$$y(t) = \left\| \mu^{d(t, s \wedge t)} \, x(s \wedge t) + v_2(t) \right\| u(t)$$

where $v_1, v_2$ are independent white noise processes; and $d(\ ,\ )$ denotes the distance between a node and one of its ancestors on the tree (e.g., $d(s, p(s)) = 1$). For nodes $s$ and $t$ at the same scale and orientation but spatially separated by a distance of $\Delta(s, t)$, the distance between $s$ and the common ancestor $s \wedge t$ grows as $d(s, s \wedge t) \sim [\log_2(\Delta(s, t)) + 1]$.

The first row of Figure 4 shows the range of behaviors seen in joint distributions taken from a wavelet subband of a particular natural image, compared to simulated GSM gamma distributions with $\mu = 0.92$. The first column corresponds to a pair of wavelet filters in quadrature phase (i.e., related by a Hilbert transform). Note that for this pair of coefficients, the contours are nearly circular, an observation that has been previously made by Zetzsche [4]. Nevertheless, these two coefficients are dependent, as shown by the multiplicative scaling in the conditional histogram of the third row. This type of scaling dependency has been extensively documented by Simoncelli [5, 9]. Analogous plots for the simulated Gamma model, with zero spatial separation are shown in rows 2 and 4. As in the image data, the contours of the joint density are very close to circular, and the conditional distribution shows a striking variance dependency.

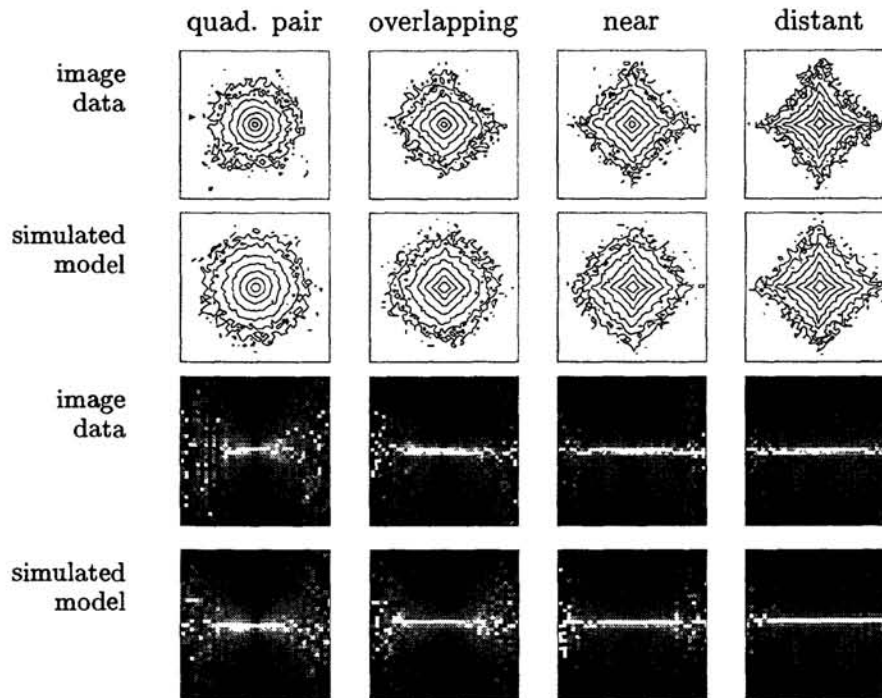

**Figure 4.** Examples of empirically observed distributions of wavelet coefficients, compared with simulated distributions from the GSM gamma model. First row: Empirical joint histograms for the "mountain" image, for four pairs of wavelet coefficients, corresponding to basis functions with spatial separations $\Delta = \{0, 4, 8, 128\}$. Second row: Simulated joint distributions for Gamma variables with $\mu = 0.92$ and the same spatial separations. Contour lines are drawn at equal intervals of log probability. Third row: Empirical conditional histograms for the "mountain" image. Fourth row: Simulated conditional histograms for Gamma variables. For these conditional distributions, intensity corresponds to probability, except that each column has been independently rescaled to fill the full range of intensities.

The remaining three columns of figure 4 show pairs of coefficients drawn from identical wavelet filters at spatial displacements $\Delta = \{4, 8, 128\}$, corresponding to a pair of overlapping filters, a pair of nearby filters, and a distant pair. Note the progression in the contour shapes from off-circular, to a diamond shape, to a concave "star" shape. The model distributions behave similarly, and show the same range of contours for simulated pairs of coefficients. Thus, consistent with empirical observations, a GSM model can produce a range of dependency between pairs of wavelet coefficients. Again, the marginal histograms retain the same form throughout this range.

## 3   Conclusions

We have proposed the class of Gaussian scale mixtures for modeling natural images. Models in this class typically exhibit heavy-tailed marginals, as well as multiplicative scaling between adjacent coefficients. We have demonstrated that a particular GSM (the symmetrized Gamma family) accounts well for both the marginal and joint distributions of wavelet coefficients from natural images. More importantly, this model suggests a hidden Markov structure for natural images, in which wavelet coefficients are linked by hidden multipliers. Romberg et al. [10] have made a related proposal using two-state discrete multipliers, corresponding to a finite mixture of Gaussians.

We have demonstrated that the hidden multipliers can be locally estimated from measurements of wavelet coefficients. Thus, by conditioning on fixed values of the multipliers, estimation problems may be reduced to the classical Gaussian case. Moreover, we described how to link the multipliers on a multiresolution tree, and showed that such a random cascade model accounts well for the drop-off in dependence of spatially separated coefficients. We are currently exploring EM-like algorithms for the problem of dual parameter and state estimation.

## Acknowledgements

We thank Bill Freeman, David Mumford, Mike Schneider, Ilya Pollak, and Alan Willsky for helpful discussions.

## Footnotes

[1]We use the steerable pyramid, an overcomplete multiscale representation described in [13]. The marginal and joint statistics of other multiscale oriented representations are similar.

## References

[1] D. L. Ruderman and W. Bialek. Statistics of natural images: Scaling in the woods. *Phys. Rev. Letters*, 73(6):814–817, 1994.

[2] D. J. Field. Relations between the statistics of natural images and the response properties of cortical cells. *J. Opt. Soc. Am. A*, 4(12):2379–2394, 1987.

[3] A. H. Tewfik and M. Kim. Correlation structure of the discrete wavelet coefficients of fractional Brownian motion. *IEEE Trans. Info. Theory*, 38:904–909, Mar. 1992.

[4] C. Zetzsche, B. Wegmann, and E. Barth. Nonlinear aspects of primary vision: Entropy reduction beyond decorrelation. In *Int'l Symp. Soc. for Info. Display*, volume 24, pages 933–936, 1993.

[5] E. P. Simoncelli. Statistical models for images: Compression, restoration and synthesis. In *31st Asilomar Conf.*, pages 673–678, Nov. 1997.

[6] S. G. Mallat. A theory for multiresolution signal decomposition: the wavelet representation. *IEEE Pat. Anal. Mach. Intell.*, 11:674–693, July 1989.

[7] E. P. Simoncelli and E. H. Adelson. Noise removal via Bayesian wavelet coring. In *Proc. IEEE ICIP*, volume I, pages 379–382, September 1996.

[8] P. Moulin and J. Liu. Analysis of multiresolution image denoising schemes using a generalized Gaussian and complexity priors. *IEEE Trans. Info. Theory*, 45:909–919, Apr. 1999.

[9] R. W. Buccigrossi and E. P. Simoncelli. Image compression via joint statistical characterization in the wavelet domain. *IEEE Trans. Image. Proc.*, 8(12):1688–1701, Dec. 1999.

[10] J.K. Romberg, H. Choi, and R.G. Baraniuk. Bayesian wavelet domain image modeling using hidden Markov trees. In *Proc. IEEE ICIP*, Kobe, Japan, Oct. 1999.

[11] J. Huang and D. Mumford. Statistics of natural images and models. In *CVPR*, paper 216, 1999.

[12] D.F. Andrews and C.L. Mallows. Scale mixtures of normal distributions. *J. Royal Stat. Soc.*, 36:99–102, 1974.

[13] E. P. Simoncelli and W. T. Freeman. The steerable pyramid: A flexible architecture for multi-scale derivative computation. In *Proc. IEEE ICIP*, volume III, pages 444–447, Oct. 1995.

[14] E. P. Simoncelli and O. Schwartz. Image statistics and cortical normalization models. In M. S. Kearns, S. A. Solla, and D. A. Cohn, editors, *Adv. Neural Information Processing Systems*, volume 11, pages 153–159, Cambridge, MA, May 1999.

[15] K. Chou, A. Willsky, and R. Nikoukhah. Multiscale systems, Kalman filters, and Riccati equations. *IEEE Trans. Automatic Control*, 39(3):479–492, Mar. 1994.
